# Bayesian binning beats approximate alternatives: estimating peristimulus time histograms

**Dominik Endres, Mike Oram, Johannes Schindelin and Peter Földiák**
School of Psychology
University of St. Andrews
KY16 9JP, UK
{dme2,mwo,js108,pf2}@st-andrews.ac.uk

## Abstract

The peristimulus time histogram (PSTH) and its more continuous cousin, the spike density function (SDF) are staples in the analytic toolkit of neurophysiologists. The former is usually obtained by binning spike trains, whereas the standard method for the latter is smoothing with a Gaussian kernel. Selection of a bin width or a kernel size is often done in an relatively arbitrary fashion, even though there have been recent attempts to remedy this situation [1, 2]. We develop an exact Bayesian, generative model approach to estimating PSTHs and demonstate its superiority to competing methods. Further advantages of our scheme include automatic complexity control and error bars on its predictions.

## 1 Introduction

Plotting a peristimulus time histogram (PSTH), or a spike density function (SDF), from spiketrains evoked by and aligned to a stimulus onset is often one of the first steps in the analysis of neurophysiological data. It is an easy way of visualizing certain characteristics of the neural response, such as instantaneous firing rates (or firing probabilities), latencies and response offsets. These measures also implicitly represent a model of the neuron's response as a function of time and are important parts of their functional description. Yet PSTHs are frequently constructed in an unsystematic manner, e.g. the choice of time bin size is driven by result expectations as much as by the data. Recently, there have been more principled approaches to the problem of determining the appropriate temporal resolution [1, 2].

We develop an exact Bayesian solution, apply it to real neural data and demonstrate its superiority to competing methods. Note that we do in no way claim that a PSTH is a complete generative description of spiking neurons. We are merely concerned with inferring that part of the generative process which can be described by a PSTH in a Bayes-optimal way.

## 2 The model

Suppose we wanted to model a PSTH on $[t_{min}, t_{max}]$, which we discretize into $T$ contiguous intervals of duration $\Delta t = (t_{max} - t_{min})/T$ (see fig.1, left). We select a discretization fine enough so that we will not observe more than one spike in a $\Delta t$ interval for any given spike train. This can be achieved easily by choosing a $\Delta t$ shorter than the absolute refractory period of the neuron under investigation. Spike train $i$ can then be represented by a binary vector $\vec{z}^i$ of dimensionality $T$. We model the PSTH by $M+1$ contiguous, non-overlapping bins having inclusive upper boundaries $k_m$, within which the firing probability $P(spike|t \in (t_{min} + \Delta t(k_{m-1}+1), t_{min} + \Delta t(k_m+1)]) = f_m$ is constant. $M$ is the number of bin boundaries inside $[t_{min}, t_{max}]$. The probability of a spike train

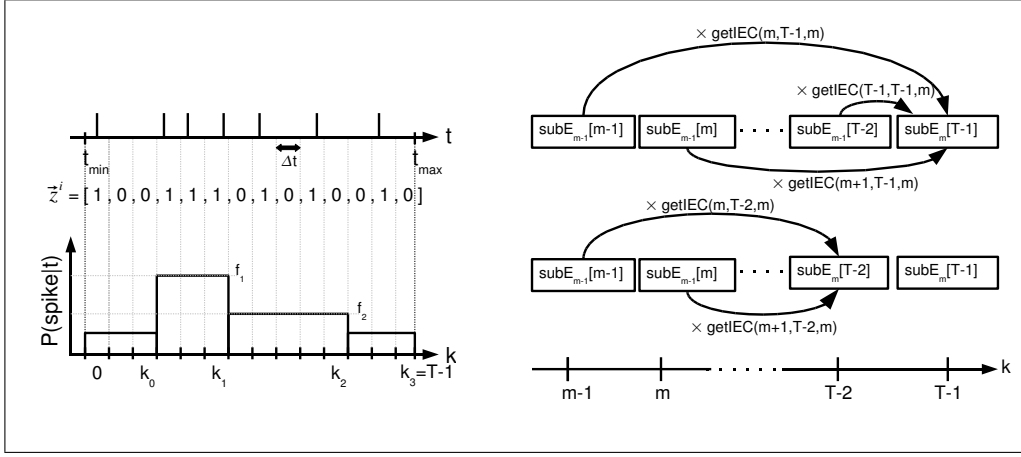

Figure 1: *Left*: Top: A spike train, recorded between times $t_{min}$ and $t_{max}$ is represented by a binary vector $\vec{z}^i$. Bottom: The time span between $t_{min}$ and $t_{max}$ is discretized into $T$ intervals of duration $\Delta t = (t_{max} - t_{min})/T$, such that interval $k$ lasts from $k \times \Delta t + t_{min}$ to $(k + 1) \times \Delta t + t_{min}$. $\Delta t$ is chosen such that at most one spike is observed per $\Delta t$ interval for any given spike train. Then, we model the firing probabilities $P(spike|t)$ by $M + 1 = 4$ contiguous, non-overlapping bins ($M$ is the number of bin boundaries inside the time span $[t_{min}, t_{max}]$), having inclusive upper boundaries $k_m$ and $P(spike|t \in (t_{min} + \Delta t(k_{m-1}+1), t_{min} + \Delta t(k_m + 1)]) = f_m$. *Right*: The core iteration. To compute the evidence contribution $\text{subE}_m[T-1]$ of a model with a bin boundary at $T-1$ and $m$ bin boundaries prior to $T-1$, we sum over all evidence contributions of models with a bin boundary at $k$ and $m-1$ bin boundaries prior to $k$, where $k \geq m-1$, because $m$ bin boundaries must occupy at least time intervals $0; \ldots ; m-1$. This takes $O(T)$ operations. Repeat the procedure to obtain $\text{subE}_m[T-2]; \ldots ; \text{subE}_m[m]$. Since we expect $T \gg m$, computing all $\text{subE}_m[k]$ given $\text{subE}_{m-1}[k]$ requires $O(T^2)$ operations. For details, see text.

$\vec{z}^i$ of independent spikes/gaps is then

$$P(\vec{z}^i|\{f_m\}, \{k_m\}, M) = \prod_{m=0}^{M} f_m^{s(\vec{z}^i, m)}(1 - f_m)^{g(\vec{z}^i, m)} \tag{1}$$

where $s(\vec{z}^i, m)$ is the number of spikes and $g(\vec{z}^i, m)$ is the number of non-spikes, or gaps in spike-train $\vec{z}^i$ in bin $m$, i.e. between intervals $k_{m-1}+1$ and $k_m$ (both inclusive). In other words, we model the spiketrains by an inhomogeneous Bernoulli process with piecewise constant probabilities. We also define $k_{-1} = -1$ and $k_M = T - 1$. Note that there is no binomial factor associated with the contribution of each bin, because we do *not* want to ignore the spike timing information within the bins, but rather, we try to build a simplified generative model of the spike train. Therefore, the probability of a (multi)set of spiketrains $\{\vec{z}^i\} = \{z_1, \ldots, z_N\}$, assuming independent generation, is

$$\begin{aligned} P(\{\vec{z}^i\}|\{f_m\}, \{k_m\}, M) &= \prod_{i=1}^{N} \prod_{m=0}^{M} f_m^{s(\vec{z}^i, m)}(1 - f_m)^{g(\vec{z}^i, m)} \\ &= \prod_{m=0}^{M} f_m^{s(\{\vec{z}^i\}, m)}(1 - f_m)^{g(\{\vec{z}^i\}, m)} \end{aligned} \tag{2}$$

where $s(\{\vec{z}^i\}, m) = \sum_{i=1}^{N} s(\vec{z}^i, m)$ and $g(\{\vec{z}^i\}, m) = \sum_{i=1}^{N} g(\vec{z}^i, m)$

## 2.1 The priors

We will make a non-informative prior assumption for $p(\{f_m\}, \{k_m\})$, namely

$$p(\{f_m\}, \{k_m\}|M) = p(\{f_m\}|M)P(\{k_m\}|M). \tag{3}$$

i.e. we have no *a priori* preferences for the firing rates based on the bin boundary positions. Note that the prior of the $f_m$, being continuous model parameters, is a density. Given the form of eqn.(1) and the constraint $f_m \in [0,1]$, it is natural to choose a conjugate prior

$$p(\{f_m\}|M) = \prod_{m=0}^{M} \mathrm{B}(f_m; \sigma_m, \gamma_m). \tag{4}$$

The Beta density is defined in the usual way [3]:

$$\mathrm{B}(p; \sigma, \gamma) = \frac{\Gamma(\sigma + \gamma)}{\Gamma(\sigma)\Gamma(\gamma)} p^{\sigma}(1-p)^{\gamma}. \tag{5}$$

There are only finitely many configurations of the $k_m$. Assuming we have no preferences for any of them, the prior for the bin boundaries becomes

$$P(\{k_m\}|M) = \frac{1}{\left(\begin{array}{c} T-1 \\ M \end{array}\right)}. \tag{6}$$

where the denominator is just the number of possibilities in which $M$ ordered bin boundaries can be distributed across $T-1$ places (bin boundary $M$ always occupies position $T-1$, see fig.1,left , hence there are only $T-1$ positions left).

# 3    Computing the evidence $P(\{\vec{z}^i\}|M)$

To calculate quantities of interest for a given $M$, e.g. predicted firing probabilities and their variances or expected bin boundary positions, we need to compute averages over the posterior

$$p(\{f_m\}, \{k_m\}|M, \{\vec{z}^i\}) = \frac{p(\{\vec{z}^i\}, \{f_m\}, \{k_m\}|M)}{P(\{\vec{z}^i\}|M)} \tag{7}$$

which requires the evaluation of the evidence, or marginal likelihood of a model with $M$ bins:

$$P(\{\vec{z}^i\}|M) = \sum_{k_{M-1}=M-1}^{T-2} \sum_{k_{M-2}=M-2}^{k_{M-1}-1} \cdots \sum_{k_0=0}^{k_1-1} P(\{\vec{z}^i\}|\{k_m\}, M) P(\{k_m\}|M) \tag{8}$$

where the summation boundaries are chosen such that the bins are non-overlapping and contiguous and

$$P(\{\vec{z}^i\}|\{k_m\}, M) = \int_0^1 df_0 \int_0^1 df_1 \ldots \int_0^1 df_M P(\{\vec{z}^i\}|\{f_m\}, \{k_m\}, M) p(\{f_m\}|M). \tag{9}$$

By virtue of eqn.(2) and eqn.(4), the integrals can be evaluated:

$$P(\{\vec{z}^i\}|\{k_m\}, M) = \prod_{m=0}^{M} \frac{\Gamma(s(\{\vec{z}^i\}, m) + \sigma_m)\Gamma(g(\{\vec{z}^i\}, m) + \gamma_m)}{\Gamma(s(\{\vec{z}^i\}, m) + \sigma_m + g(\{\vec{z}^i\}, m) + \gamma_m)} \prod_{m=0}^{M} \frac{\Gamma(\sigma_m + \gamma_m)}{\Gamma(\sigma_m)\Gamma(\gamma_m)}. \tag{10}$$

Computing the sums in eqn.(8) quickly is a little tricky. A naïve approach would suggest that a computational effort of $O(T^M)$ is required. However, because eqn.(10) is a product with one factor per bin, and because each factor depends only on spike/gap counts and prior parameters in that bin, the process can be expedited. We will use an approach very similar to that described in [4, 5] in the context of density estimation and in [6, 7] for Bayesian function approximation: define the function

$$\mathrm{getIEC}(k_s, k_e, m) := \frac{\Gamma(s(\{\vec{z}^i\}, k_s, k_e) + \sigma_m)\Gamma(g(\{\vec{z}^i\}, k_s, k_e) + \gamma_m)}{\Gamma(s(\{\vec{z}^i\}, k_s, k_e) + \sigma_m + g(\{\vec{z}^i\}, k_s, k_e) + \gamma_m)} \tag{11}$$

where $s(\{\vec{z}^i\}, k_s, k_e)$ is the number of spikes and $g(\{\vec{z}^i\}, k_s, k_e)$ is the number of gaps in $\{\vec{z}^i\}$ between the start interval $k_s$ and the end interval $k_e$ (both included). Furthermore, collect all contributions to eqn.(8) that do not depend on the data (i.e. $\{\vec{z}^i\}$) and store them in the array $\mathrm{pr}[M]$:

$$\mathrm{pr}[M] := \frac{\prod_{m=0}^{M} \frac{\Gamma(\sigma_m + \gamma_m)}{\Gamma(\sigma_m)\Gamma(\gamma_m)}}{\left(\begin{array}{c} T-1 \\ M \end{array}\right)}. \tag{12}$$

Substituting eqn.(10) into eqn.(8) and using the definitions (11) and (12), we obtain

$$P(\{\vec{z}^i\}|M) \propto \sum_{k_{M-1}=M-1}^{T-2} \cdots \sum_{k_0=0}^{k_1-1} \prod_{m=1}^{M} \text{getIEC}(k_{m-1}+1, k_m, m)\text{getIEC}(0, k_0, 0) \qquad (13)$$

with $k_M = T - 1$ and the constant of proportionality being $\text{pr}[M]$. Since the factors on the r.h.s. depend only on two consecutive bin boundaries each, it is possible to apply dynamic programming [8]: rewrite the r.h.s. by 'pushing' the sums as far to the right as possible:

$$P(\{\vec{z}^i\}|M) \propto \sum_{k_{M-1}=M-1}^{T-2} \text{getIEC}(k_{M-1}+1, T-1, M) \sum_{k_{M-2}=M-2}^{k_{M-1}-1} \text{getIEC}(k_{M-2}+1, k_{M-1}, M-1)$$

$$\times \cdots \sum_{k_0=0}^{k_1-1} \text{getIEC}(k_0+1, k_1, 1)\text{getIEC}(0, k_0, 0). \qquad (14)$$

Evaluating the sum over $k_0$ requires $O(T)$ operations (assuming that $T \gg M$, which is likely to be the case in real-world applications). As the summands depend also on $k_1$, we need to repeat this evaluation $O(T)$ times, i.e. summing out $k_0$ for all possible values of $k_1$ requires $O(T^2)$ operations. This procedure is then repeated for the remaining $M - 1$ sums, yielding a total computational effort of $O(MT^2)$. Thus, initialize the array $\text{subE}_0[k] := \text{getIEC}(0, k, 0)$, and iterate for all $m = 1, \ldots, M$:

$$\text{subE}_m[k] := \sum_{r=m-1}^{k-1} \text{getIEC}(r+1, k, m)\text{subE}_{m-1}[r], \qquad (15)$$

A close look at eqn.(14) reveals that while we sum over $k_{M-1}$, we need $\text{subE}_{M-1}[k]$ for $k = M - 1; \ldots; T - 2$ to compute the evidence of a model with its latest boundary at $T - 1$. We can, however, compute $\text{subE}_{M-1}[T-1]$ with little extra effort, which is, up to a factor $\text{pr}[M-1]$, equal to $P(\{\vec{z}^i\}|M-1)$, i.e. the evidence for a model with $M - 1$ bin boundaries. Moreover, having computed $\text{subE}_m[k]$, we do not need $\text{subE}_{m-1}[k-1]$ anymore. Hence, the array $\text{subE}_{m-1}[k]$ can be reused to store $\text{subE}_m[k]$, if overwritten in reverse order. In pseudo-code ($E[m]$ contains the evidence of a model with $m$ bin boundaries inside $[t_{min}, t_{max}]$ after termination):

Table 1: Computing the evidences of models with up to $M$ bin boundaries

1. for $k := 0 \ldots T - 1$ : $\text{subE}[k] := \text{getIEC}(0, k, 0)$
2. $E[0] := \text{subE}[T-1] \times \text{pr}[0]$
3. for $m := 1 \ldots M$ :
   (a) if $m = M$ then $l := T - 1$ else $l := m$
   (b) for $k := T - 1 \ldots l$
       $\text{subE}[k] := \sum_{r:=m-1}^{k-1} \text{subE}[r] \times \text{getIEC}(r+1, k, m)$
   (c) $E[m] = \text{subE}[T-1] \times \text{pr}[m]$
4. return $E[]$

## 4 Predictive firing rates and variances

We will now calculate the predictive firing rate $P(spike|\tilde{k}, \{\vec{z}^i\}, M)$. For a given configuration of $\{f_m\}$ and $\{k_m\}$, we can write

$$P(spike|\tilde{k}, \{f_m\}, \{k_m\}, M) = \sum_{m=0}^{M} f_m \mathbf{1}(\tilde{k} \in \{k_{m-1}+1, k_m\}) \qquad (16)$$

where the indicator function $\mathbf{1}(x) = 1$ iff $x$ is true and 0 otherwise. Note that the probability of a spike given $\{k_m\}$ and $\{f_m\}$ does not depend on any observed data. Since the bins are non-overlapping, $\tilde{k} \in \{k_{m-1}+1, k_m\}$ is true for exactly one summand and $P(spike|\tilde{k}, \{\vec{z}^i\}, \{k_m\})$ evaluates to the corresponding firing rate.

To finish we average eqn.(16) over the posterior eqn.(7). The denominator of eqn.(7) is independent of $\{f_m\}, \{k_m\}$ and is obtained by integrating/summing the numerator via the algorithm in table 1. Thus, we only need to multiply the integrand of eqn.(9) (i.e. the numerator of the posterior) with $P(spike|\tilde{k}, \{f_m\}, \{k_m\}, M)$, thereby replacing eqn.(11) with

$$\text{getIEC}(k_s, k_e, m) := \frac{\Gamma(s(\{\vec{z}^i\}, k_s, k_e) + \mathbf{1}(\tilde{k} \in \{k_s, k_e\}) + \sigma_m)\Gamma(g(\{\vec{z}^i\}, k_s, k_e) + \gamma_m)}{\Gamma(s(\{\vec{z}^i\}, k_s, k_e) + \mathbf{1}(\tilde{k} \in \{k_s, k_e\}) + \sigma_m + g(\{\vec{z}^i\}, k_s, k_e) + \gamma_m)} \quad (17)$$

i.e. we are adding an additional spike to the data at $\tilde{k}$. Call the array returned by this modified algorithm $\text{E}_{\tilde{k}}[]$. By virtue of eqn.(7) we then find $P(spike|\tilde{k}, \{\vec{z}^i\}, M) = \frac{\text{E}_{\tilde{k}}[M]}{\text{E}[M]}$. To evaluate the variance, we need the posterior expectation of $f_m^2$. This can be computed by adding two spikes at $\tilde{k}$.

## 5 Model selection vs. model averaging

To choose the best $M$ given $\{\vec{z}^i\}$, or better, a probable range of $M$s, we need to determine the model posterior

$$P(M|\{\vec{z}^i\}) = \frac{P(\{\vec{z}^i\}|M)P(M)}{\sum_m P(\{\vec{z}^i\}|m)P(m)} \quad (18)$$

where $P(M)$ is the prior over $M$, which we assume to be uniform. The sum in the denominator runs over all values of $m$ which we choose to include, at most $0 \leq m \leq T - 1$.

Once $P(M|\{\vec{z}^i\})$ is evaluated, we could use it to select the most probable $M'$. However, making this decision means 'contriving' information, namely that all of the posterior probability is concentrated at $M'$. Thus we should rather average any predictions over all possible $M$, even if evaluating such an average has a computational cost of $O(T^3)$, since $M \leq T - 1$. If the structure of the data allow, it is possible, and useful given a large enough $T$, to reduce this cost by finding a range of $M$, such that the risk of excluding a model even though it provides a good description of the data is low. In analogy to the significance levels of orthodox statistics, we shall call this risk $\alpha$. If the posterior of $M$ is unimodal (which it has been in most observed cases, see fig.3, right, for an example), we can then choose the smallest interval of $M$s around the maximum of $P(M|\{\vec{z}^i\})$ such that

$$P(M_{min} \leq M \leq M_{max}|\{\vec{z}^i\}) \leq 1 - \alpha \quad (19)$$

and carry out the averages over this range of $M$ after renormalizing the model posterior.

## 6 Examples and comparison to other methods

### 6.1 Data acquisition

We obtained data through [9], where the experimental protocols have been described. Briefly, extra-cellular single-unit recordings were made using standard techniques from the upper and lower banks of the anterior part of the superior temporal sulcus (STSa) and the inferior temporal cortex (IT) of two monkeys (Macaca mulatta) performing a visual fixation task. Stimuli were presented for 333 ms followed by an 333 ms inter-stimulus interval in random order. The anterior-posterior extent of the recorded cells was from 7mm to 9mm anterior of the interaural plane consistent with previous studies showing visual responses to static images in this region [10, 11, 12, 13]. The recorded cells were located in the upper bank (TAa, TPO), lower bank (TEa, TEm) and fundus (PGa, IPa) of STS and in the anterior areas of TE (AIT of [14]). These areas are rostral to FST and we collectively call them the anterior STS (STSa), see [15] for further discussion. The recorded firing patters were turned into distinct samples, each of which contained the spikes from $-300$ ms before to 600 ms after the stimulus onset with a temporal resolution of 1 ms.

### 6.2 Inferring PSTHs

To see the method in action, we used it to infer a PSTH from 32 spiketrains recorded from one of the available STSa neurons (see fig.2, A). Spikes times are relative to the stimulus onset. We discretized the interval from $-100ms$ pre-stimulus to $500ms$ post-stimulus into $\Delta t = 1$ms time intervals and

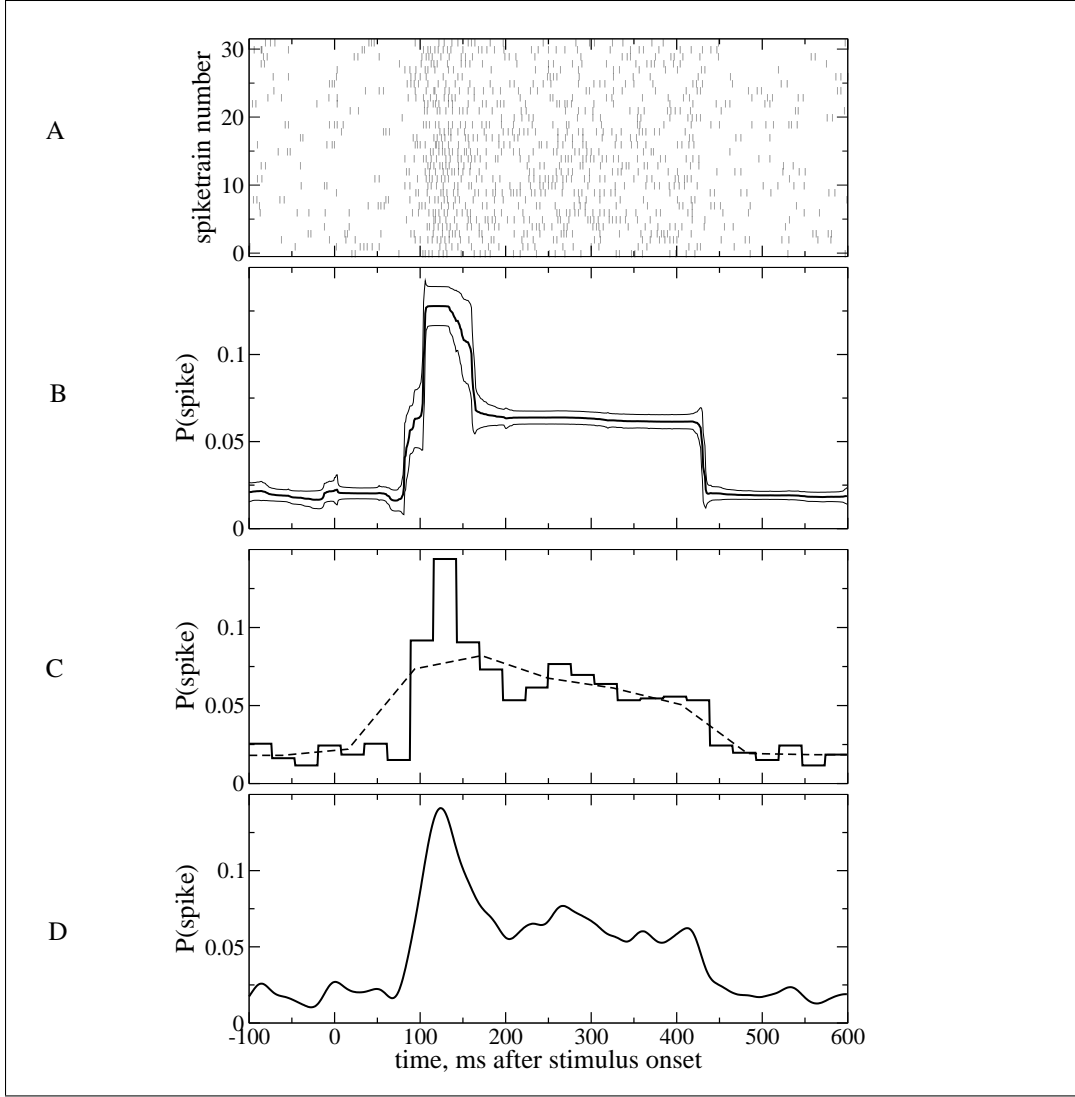

Figure 2: Predicting a PSTH/SDF with 3 different methods. *A*: the dataset used in this comparison consisted of 32 spiketrains recorded from a STSa neuron. Each tick mark represents a spike. *B*: PSTH inferred with our Bayesian binning method. The thick line represents the predictive firing rate (section 4), the thin lines show the predictive firing rate $\pm 1$ standard deviation. Models with $4 \leq M \leq 13$ were included on a risk level of $\alpha = 0.1$ (see eqn.(19)). *C*: bar PSTH (solid lines), optimal binsize $\approx$ 26ms, and line PSTH (dashed lines), optimal binsize $\approx$ 78ms, computed by the methods described in [1, 2]. *D*: SDF obtained by smoothing the spike trains with a 10ms Gaussian kernel.

computed the model posterior (eqn.(18)) (see fig.3, right). The prior parameters were equal for all bins and set to $\sigma_m = 1$ and $\gamma_m = 32$. This choice corresponds to a firing probability of $\approx 0.03$ in each 1 ms time interval (30 spikes/s), which is typical for the neurons in this study[1]. Models with $4 \leq M \leq 13$ (expected bin sizes between $\approx$ 23ms-148ms) were included on an $\alpha = 0.1$ risk level (eqn.(19)) in the subsequent calculation of the predictive firing rate (i.e. the *expected* firing rate, hence the continuous appearance) and standard deviation (fig.2, B). Fig.2, C, shows a bar PSTH and a line PSTH computed with the recently developed methods described in [1, 2]. Roughly speaking,

these methods try to optimize a compromise between minimal within-bin variance and maximal between-bin variance. In this example, the bar PSTH consists of 26 bins. Graph D in fig.2 depicts a SDF obtained by smoothing the spiketrains with a 10ms wide Gaussian kernel, which is a standard way of calculating SDFs in the neurophysiological literature.

All tested methods produce results which are, upon cursory visual inspection, largely consistent with the spiketrains. However, Bayesian binning is better suited than Gaussian smoothing to model steep changes, such as the transient response starting at $\approx$ 100ms. While the methods from [1, 2] share this advantage, they suffer from two drawbacks: firstly, the bin boundaries are evenly spaced, hence the peak of the transient is later than the scatterplots would suggest. Secondly, because the bin duration is the only parameter of the model, these methods are forced to put many bins even in intervals that are relatively constant, such as the baselines before and after the stimulus-driven response. In contrast, Bayesian binning, being able to put bin boundaries anywhere in the time span of interest, can model the data with less bins – the model posterior has its maximum at $M = 6$ (7 bins), whereas the bar PSTH consists of 26 bins.

## 6.3 Performance comparison

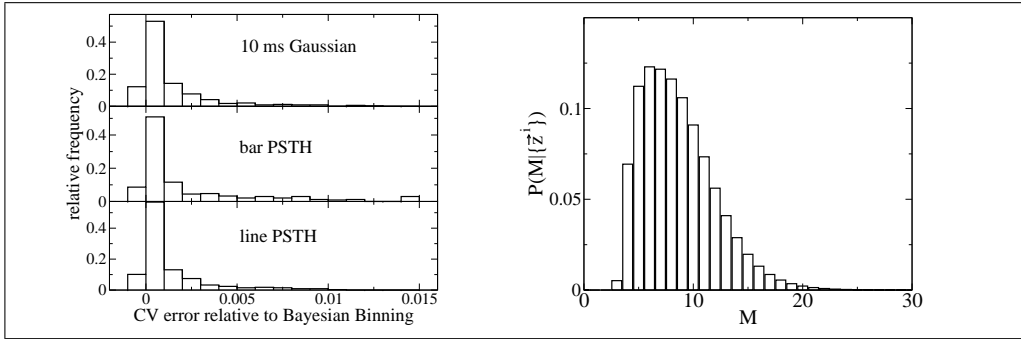

Figure 3: *Left*: Comparison of Bayesian Binning with competing methods by 5-fold crossvalidation. The CV error is the negative expected log-probability of the test data. The histograms show relative frequencies of CV error differences between 3 competing methods and our Bayesian binning approach. Gaussian: SDFs obtained by Gaussian smoothing of the spiketrains with a 10 ms kernel. Bar PSTH and line PSTH: PSTHs computed by the binning methods described in [1, 2]. *Right*: Model posterior $P(M|\{\vec{z}^i\})$ (see eqn.(18)) computed from the data shown in fig.2. The shape is fairly typical for model posteriors computed from the neural data used in this paper: a sharp rise at a moderately low $M$ followed by a maximum (here at $M = 6$) and an approximately exponential decay. Even though a maximum $M$ of 699 would have been possible, $P(M > 23|\{\vec{z}^i\}) < 0.001$. Thus, we can accelerate the averaging process for quantities of interest (e.g. the predictive firing rate, section 4) by choosing a moderately small maximum $M$.

For a more rigorous method comparison, we split the data into distinct sets, each of which contained the responses of a cell to a different stimulus. This procedure yielded 336 sets from 20 cells with at least 20 spiketrains per set. We then performed 5-fold crossvalidation, the crossvalidation error is given by the negative logarithm of the data (spike or gap) in the test sets:

$$\text{CV error} = - \langle \log(P(spike|t)) \rangle . \tag{20}$$

Thus, we measure how well the PSTHs predict the test data. The Gaussian SDFs were discretized into 1 ms time intervals prior to the procedure. We average the CV error over the 5 estimates to obtain a single estimate for each of the 336 neuron/stimulus combinations. On average, the negative log likelihood of our Bayesian approach predicting the test data ($0.04556 \pm 0.00029$, mean $\pm$ SEM) was significantly better than any of the other methods (10ms Gaussian kernel: $0.04654 \pm 0.00028$; Bar PSTH: $0.04739 \pm 0.00029$; Line PSTH: $0.04658 \pm 0.00029$). To directly compare the performance of different methods we calculate the difference in the CV error for each neuron/stimulus combination. Here a positive value indicates that Bayesian binning predicts the test data more accurately than the alternative method. Fig.3, left, shows the relative frequencies of CV error differences between the 3 other methods and our approach. Bayesian binning predicted the data better than the three other

methods in at least 295/336 cases, with a minimal difference of $\approx -0.0008$, indicating the general utility of this approach.

## 7 Summary

We have introduced an exact Bayesian binning method for the estimation of PSTHs. Besides treating uncertainty – a real problem with small neurophysiological datasets – in a principled fashion, it also outperforms competing methods on real neural data. It offers automatic complexity control because the model posterior can be evaluated. While its computational cost is significantly higher than that of the methods we compared it to, it is still fast enough to be useful: evaluating the predictive probability takes less than 1s on a modern PC[2], with a small memory footprint ($<$10MB for 512 spiketrains).

Moreover, our approach can easily be adapted to extract other characteristics of neural responses in a Bayesian way, e.g. response latencies or expected bin boundary positions. Our method reveals a clear and sharp initial response onset, a distinct transition from the transient to the sustained part of the response and a well-defined offset. An extension towards joint PSTHs from simultaneous multi-cell recordings is currently being implemented.

## Footnotes

[1]Alternatively, one could search for the $\sigma_m, \gamma_m$ which maximize of $P(\{\vec{z}^i\}|\sigma_m, \gamma_m) = \sum_M P(\{\vec{z}^i\}|M)P(M|\sigma_m, \gamma_m)$, where $P(\{\vec{z}^i\}|M)$ is given by eqn.(8). Using a uniform $P(M|\sigma_m, \gamma_m)$, we found $\sigma_m \approx 2.3$ and $\gamma_m \approx 37$ for the data in fig.2, A

[2]3.2 GHz Intel Xeon$^{\text{TM}}$, SuSE Linux 10.0

## References

[1] H. Shimazaki and S. Shinomoto. A recipe for optimizing a time-histogram. In B. Schölkopf, J. Platt, and T. Hoffman, editors, *Advances in Neural Information Processing Systems 19*, pages 1289–1296. MIT Press, Cambridge, MA, 2007.

[2] H. Shimazaki and S. Shinomoto. A method for selecting the bin size of a time histogram. *Neural Computation*, 19(6):1503–1527, 2007.

[3] J.O. Berger. *Statistical Decision Theory and Bayesian Analysis*. Springer, New York, 1985.

[4] D. Endres and P. Földiák. Bayesian bin distribution inference and mutual information. *IEEE Transactions on Information Theory*, 51(11), 2005.

[5] D. Endres. *Bayesian and Information-Theoretic Tools for Neuroscience*. PhD thesis, School of Psychology, University of St. Andrews, U.K., 2006. http://hdl.handle.net/10023/162.

[6] M. Hutter. Bayesian regression of piecewise constant functions. Technical Report arXiv:math/0606315v1, IDSIA-14-05, 2006.

[7] M. Hutter. Exact bayesian regression of piecewise constant functions. *Journal of Bayesian Analysis*, 2(4):635–664, 2007.

[8] D. P. Bertsekas. *Dynamic Programming and Optimal Control*. Athena Scientific, 2000.

[9] M. W. Oram, D. Xiao, B. Dritschel, and K.R. Payne. The temporal precision of neural signals: A unique role for response latency? *Philosophical Transactions of the Royal Society, Series B*, 357:987–1001, 2002.

[10] CJ Bruce, R Desimone, and CG Gross. Visual properties of neurons in a polysensory area in superior temporal sulcus of the macaque. *Journal of Neurophysiology*, 46:369–384, 1981.

[11] DI Perrett, ET Rolls, and W Caan. Visual neurons responsive to faces in the monkey temporal cortex. *Expl. Brain. Res.*, 47:329–342, 1982.

[12] G.C. Baylis, E.T. Rolls, and C.M. Leonard. Functional subdivisions of the temporal lobe neocortex. 1987.

[13] M. W. Oram and D. I. Perrett. Time course of neural responses discriminating different views of the face and head. *Journal of Neurophysiology*, 68(1):70–84, 1992.

[14] K Tanaka, H Saito, Y Fukada, and M Moriya. Coding visual images of objects in the inferotemporal cortex of the macaque monkey. *Journal of Neurophysiology*, pages 170–189, 1991.

[15] N.E. Barraclough, D. Xiao, C.I. Baker, M.W. Oram, and D.I. Perrett. Integration of visual and auditory information by superior temporal sulcus neurons responsive to the sight of actions. *Journal of Cognitive Neuroscience*, 17, 2005.

